# Separating Style and Content

**Joshua B. Tenenbaum**
Dept. of Brain and Cognitive Sciences
Massachusetts Institute of Technology
Cambridge, MA 02139
jbt@psyche.mit.edu

**William T. Freeman**
MERL, Mitsubishi Electric Res. Lab.
201 Broadway
Cambridge, MA 02139
freeman@merl.com

## Abstract

We seek to analyze and manipulate two factors, which we call style and content, underlying a set of observations. We fit training data with bilinear models which explicitly represent the two-factor structure. These models can adapt easily during testing to new styles or content, allowing us to solve three general tasks: *extrapolation* of a new style to unobserved content; *classification* of content observed in a new style; and *translation* of new content observed in a new style. For classification, we embed bilinear models in a probabilistic framework, *Separable Mixture Models (SMMs)*, which generalizes earlier work on factorial mixture models [7, 3]. Significant performance improvement on a benchmark speech dataset shows the benefits of our approach.

## 1 Introduction

In many pattern analysis or synthesis tasks, the observed data are generated from the interaction of two underlying factors which we will generically call "style" and "content." For example, in a character recognition task, we might observe different letters in different fonts (see Fig. 1); with handwriting, different words in different writing styles; with speech, different phonemes in different accents; with visual images, the faces of different people under different lighting conditions.

Such data raises a number of learning problems. Extracting a hidden two-factor structure given only the raw observations has received significant attention [7, 3], but unsupervised *factorial* learning of this kind has yet to prove tractable for our focus: real-world data with subtly interacting factors. We work in a more supervised setting, where labels for style or content may be available during training or testing. Figure 1 shows three problems we want to solve. Given a labelled training set of observations in multiple styles, we want to *extrapolate* a new style to unobserved content classes (Fig. 1a), *classify* content observed in a new style (Fig. 1b), and *translate* new content observed in a new style (Fig. 1c).

This paper treats these problems in a common framework, by fitting the training data with a separable model that can easily adapt during testing to new styles or content classes. We write an observation vector in style $s$ and content class $c$ as $\mathbf{y}^{sc}$. We seek to fit these observations with some model

$$\mathbf{y}^{sc} = f(\mathbf{a}^s, \mathbf{b}^c; W), \qquad (1)$$

where a particular functional form of $f$ is assumed. We must estimate parameter vectors $\mathbf{a}^s$ and $\mathbf{b}^c$ describing style $s$ and content $c$, respectively, and $W$, parameters

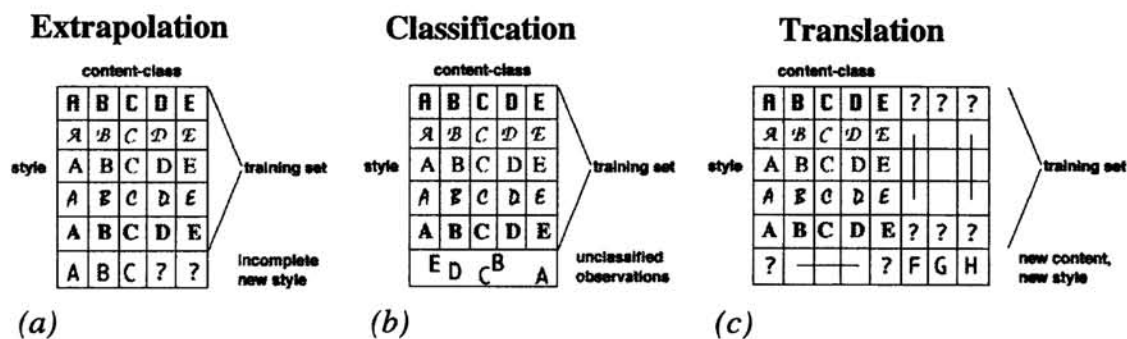

Figure 1: Given observations of content (letters) in different styles (fonts), we want to extrapolate, classify, and translate observations from a new style or content class.

for $f$ that are independent of style and content but govern their interaction. In terms of Fig. 1 (and in the spirit of [8]), the model represents what the elements of each row have in common independent of column ($\mathbf{a}^s$), what the elements of each column have in common independent of row ($\mathbf{b}^c$), and what all elements have in common independent of row and column ($W$). With these three modular components, we can solve problems like those illustrated in Fig. 1. For example, we can extrapolate a new style to unobserved content classes (Fig. 1a) by combining content and interaction parameters learned during training with style parameters estimated from available data in the new style.

## 2 Bilinear models

We propose to separate style and content using bilinear models – two-factor models that are linear in either factor when the other is held constant. These simple models are still complex enough to model subtle interactions of style and content. The empirical success of linear models in many pattern recognition applications with single-factor data (e.g. "eigenface" models of faces under varying identity but constant illumination and pose [15], or under varying illumination but constant identity and pose [5] ), makes bilinear models a natural choice when two such factors vary independently across the data set. Also, many of the computationally desirable properties of linear models extend to bilinear models. Model fitting (discussed in Section 3 below) is easy, based on efficient and well-known techniques such as the singular value decomposition (SVD) and the expectation-maximization (EM) algorithm. Model complexity can be controlled by varying model dimensionality to achieve a compromise between reproduction of the training data and generalization during testing. Finally, the approach extends to multilinear models [10], for data generated by three or more interacting factors.

We have explored two bilinear models for Eq. 1. In the *symmetric* model (so called because it treats the two factors symmetrically), we assume $f$ is a bilinear mapping given by

$$y_k^{sc} = \mathbf{a}^{s^\mathrm{T}} \mathbf{W}_k \mathbf{b}^c = \sum_{ij} a_i^s b_j^c W_{ijk}. \tag{2}$$

The $W_{ijk}$ parameters represent a set of basis functions independent of style and content, which characterize the interaction between these two factors. Observations in style $s$ and content $c$ are generated by mixing these basis functions with coefficients

given by the tensor product of $\mathbf{a}^s$ and $\mathbf{b}^c$ vectors. The model exactly reproduces the observations when the dimensionalities of $\mathbf{a}^s$ and $\mathbf{b}^c$ equal the number of styles $N_s$ and content classes $N_c$ observed. It finds coarser but more compact representations as these dimensionalities are decreased.

Sometimes it may not be practical to represent both style and content with low-dimensional vectors. For example, a linear combination of a few basis styles learned during training may not describe new styles well. We can obtain more flexible, *asymmetric* bilinear models by letting the basis functions $W_{ijk}$ themselves depend on style or content. For example, if the basis functions are allowed to depend on style, the bilinear model from Eq. 2 becomes $y_k^{sc} = \sum_{ij} a_i^s b_j^c W_{ijk}^s$. This simplifies to $y_k^{sc} = \sum_j A_{jk}^s b_j^c$, by summing out the $i$ index and identifying $A_{jk}^s \equiv \sum_i a_i^s W_{ijk}^s$. In vector notation, we have

$$\mathbf{y}^{sc} = \mathbf{A}^s \mathbf{b}^c, \tag{3}$$

where $\mathbf{A}^s$ is a matrix of basis functions specific to style $s$ (independent of content), and $\mathbf{b}^c$ is a vector of coefficients specific to content $c$ (independent of style). Alternatively, the basis functions may depend on content, which gives

$$\mathbf{y}^{sc} = \mathbf{B}^c \mathbf{a}^s. \tag{4}$$

Asymmetric models do not parameterize the rendering function $f$ independently of style and content, and so cannot translate across both factors simultaneously (Fig. 1c). Further, a matrix representation of style or content may be *too* flexible and overfit the training data. But if overfitting is not a problem or can be controlled by some additional constraint, asymmetric models may solve extrapolation and classification tasks using less training data than symmetric models.

Figure 2 illustrates an example of an asymmetric model used to separate style and content. We have collected a small database of face images, with 11 different people (content classes) in 15 different head poses (styles). The images are 22 × 32 pixels, which we treat as 704-dimensional vectors. A subset of the data is shown in Fig. 2a. Fig. 2b schematically depicts an asymmetric bilinear model of the data, with each pose represented by a set of basis vectors $\mathbf{A}^{pose}$ (shown as images) and each person represented by a set of coefficients $\mathbf{b}^{person}$. To render an image of a particular person in a particular pose, the pose-specific basis vectors are mixed according to the person-specific coefficients. Note that the basis vectors for each pose look like eigenfaces [15] in the appropriate style of each pose. However, the bilinear structure of the model ensures that corresponding basis vectors play corresponding roles across poses (e.g. the first vector holds (roughly) the mean face for that pose, the second may modulate overall head size, the third may modulate head thickness, etc.), which is crucial for adapting to new styles or content classes.

## 3   Model fitting

All the tasks shown in Fig. 1 break down into a training phase and a testing phase; both involve some model fitting. In the training phase (corresponding to the first 5 rows and columns of Figs. 1a-c), we learn all the parameters of a bilinear model from a complete matrix of observations of $N_c$ content classes in $N_s$ styles. In the testing phase (corresponding to the final rows of Figs. 1a,b and the final row and last 3 columns of Fig. 1c), we adapt the same model to data in a new style or content class (or both), estimating new parameters for the new style or content, clamping the other parameters. Then new and old parameters are combined to accomplish the desired classification, extrapolation, or translation task. This section focuses on the asymmetric model and its use in extrapolation and classification. Training and

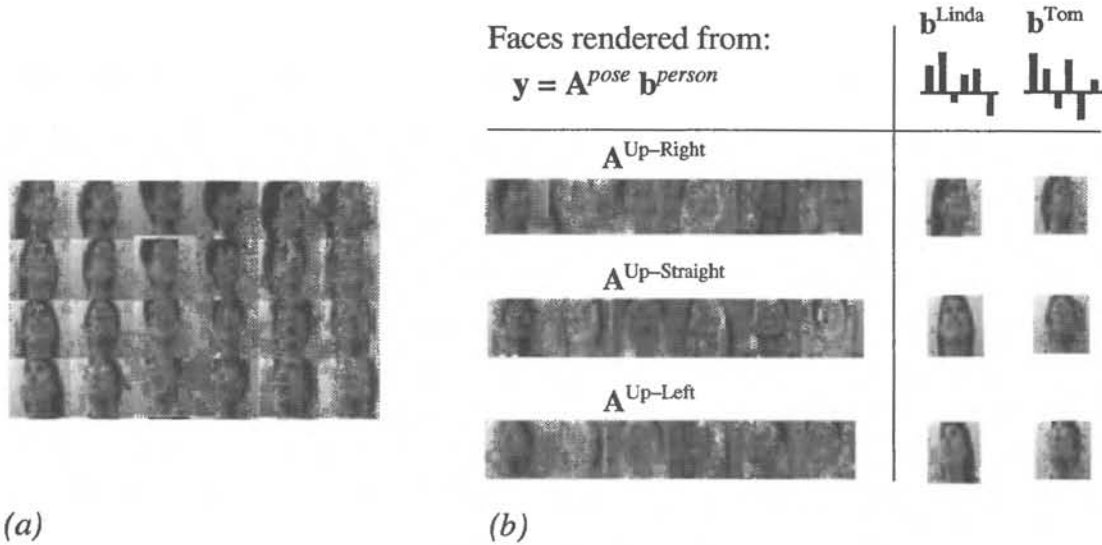

Figure 2: An illustraton of the asymmetric model, with faces varying in identity and head pose.

adaptation procedures for the symmetric model are similar and based on algorithms in [10, 11]. In [2], we describe these procedures and their application to extrapolation and translation tasks.

### 3.1  Training

Let $n_{sc}$ be the number of observations in style $s$ and content $c$, and let $\mathbf{m}^{sc} = \sum \mathbf{y}^{sc}$ be the sum of these observations. Then estimates of $\mathbf{A}^s$ and $\mathbf{b}^c$ that minimize the sum-of-squared-errors for the asymmetric model in Eq. 3 can be found by iterating the fixed point equations

$$\hat{\mathbf{A}}^s = \left[\sum_c \mathbf{m}^{sc}\mathbf{b}^{c\mathrm{T}}\right]\left[\sum_c n_{sc}\mathbf{b}^c\mathbf{b}^{c\mathrm{T}}\right]^{-1}, \hat{\mathbf{b}}^c = \left[\sum_s n_{sc}\mathbf{A}^{s\mathrm{T}}\mathbf{A}^s\right]^{-1}\left[\sum_s \mathbf{A}^{s\mathrm{T}}\mathbf{m}^{sc}\right] \quad (5)$$

obtained by setting derivatives of the error equal to 0. To ensure stability, we update the parameters according to $\mathbf{A}^s = (1-\eta)\mathbf{A}^s+\eta\hat{\mathbf{A}}^s$ and $\mathbf{b}^c = (1-\eta)\mathbf{b}^c+\eta\hat{\mathbf{b}}^c$, typically using a stepsize $0.2 < \eta < 0.5$. Replacing $\mathbf{A}^s$ with $\mathbf{B}^c$ and $\mathbf{b}^c$ with $\mathbf{a}^s$ yields the analogous procedure for training the model in Eq. 4.

If the same number of observations are available for all style-content pairs, there exists a closed-form procedure to fit the asymmetric model using the SVD. Let the $K$-dimensional vector $\bar{\mathbf{y}}^{sc}$ denote the mean of the observed data generated by style $s$ and content $c$, and stack these vectors into a single $(K \times N_s) \times N_c$ matrix

$$\mathbf{Y} = \begin{bmatrix} \bar{\mathbf{y}}^{11} & \cdots & \bar{\mathbf{y}}^{1N_c} \\ \vdots & \ddots & \\ \bar{\mathbf{y}}^{N_s 1} & & \bar{\mathbf{y}}^{N_s N_c} \end{bmatrix}. \quad (6)$$

We compute the SVD of $\mathbf{Y} = \mathbf{USV}^T$, and define the $(K \times N_s) \times J$ matrix $\mathbf{A}$ to be the first $J$ columns of $\mathbf{U}$, and the $J \times N_c$ matrix $\mathbf{B}$ to be the first $J$ rows of $\mathbf{SV}^T$. Finally, we identify $\mathbf{A}$ and $\mathbf{B}$ as the desired parameter estimates in stacked form

(see also [9, 14]),

$$\mathbf{A} = \begin{bmatrix} \mathbf{A}^1 \\ \vdots \\ \mathbf{A}^{N_s} \end{bmatrix}, \quad \mathbf{B} = \begin{bmatrix} \mathbf{b}^1 \cdots \mathbf{b}^{N_c} \end{bmatrix}. \tag{7}$$

The model dimensionality $J$ can be chosen in various standard ways: by a priori considerations, by requiring a sufficiently good approximation to the data (as measured by mean squared error or some more subjective metric), or by looking for a gap in the singular value spectrum.

## 3.2  Testing

It is straightforward to adapt the asymmetric model to an incomplete new style $s^*$, in order to extrapolate that style to unseen content. We simply estimate $\mathbf{A}^{s^*}$ from Eq. 5, using $\mathbf{b}^c$ values learned during training and restricting the sums over $c$ to those content classes observed in the new style. Then data in content $c$ and style $s^*$ can be synthesized from $\mathbf{A}^{s^*}\mathbf{b}^c$. Extrapolating incomplete new content to unseen styles is done similarly.

Adapting the asymmetric model for classification in new styles is more involved, because the content class of the new data (and possibly its style as well) is unlabeled. To deal with this uncertainty, we embed the bilinear model within a gaussian mixture model to yield a *separable mixture model (SMM)*, which can then be fit efficiently to data in new styles using the EM algorithm. Specifically, we assume that the probability of a new, unlabeled observation $\mathbf{y}$ being generated by style $s$ and content $c$ is given by a spherical gaussian centered at the prediction of the asymmetric bilinear model: $p(\mathbf{y}|s,c) \propto \exp\{-\|\mathbf{y} - \mathbf{A}^s\mathbf{b}^c\|^2/(2\sigma^2)\}$. The total probability of $\mathbf{y}$ is then $p(\mathbf{y}) = \sum_{s,c} p(\mathbf{y}|s,c)p(s,c)$; we use equal priors $p(s,c)$. We assume that the content vectors $\mathbf{b}^c$ are known from training, but that new style matrices $\mathbf{A}^s$ must be found to explain the test data. The EM algorithm alternates between computing soft style and content-class assignments $p(s,c|\mathbf{y}) = p(\mathbf{y}|s,c)p(s,c)/p(\mathbf{y})$ for each test vector $y$ given the current style matrix estimates (E-step), and estimating new style matrices by setting $\mathbf{A}^s$ to maximize $\sum_{\mathbf{y}} \log p(\mathbf{y})$ (M-step). The M-step is solved in closed form using the update rule for $\mathbf{A}^s$ from Eq. 5, with $\mathbf{m}^{sc} = \sum_{\mathbf{y}} p(s,c|\mathbf{y})\,\mathbf{y}$ and $n_{sc} = \sum_{\mathbf{y}} p(s,c|\mathbf{y})$. Test vectors in new styles can now be classified by grouping each vector $\mathbf{y}$ with the content class $c$ that maximizes $p(c|\mathbf{y}) = \sum_s p(s,c|\mathbf{y})$.

# 4  Application: speaker-adaptive speech recognition

This example illustrates our approach to style-adaptive classification on a real-world data set that is a benchmark for many connectionist learning algorithms. The data consist of 6 samples of each of 11 vowels uttered by 15 speakers of British English (originally collected by David Deterding, from the CMU neural-bench ftp archive). Each data vector consists of 10 parameters computed from a linear predictive analysis of the digitized speech. Robinson [13] compared many learning algorithms trained to categorize vowels from the first 8 speakers (4 male and 4 female) and tested on samples from the remaining 7 speakers (4 male and 3 female).

Using the SVD-based procedure described above, we fit an asymmetric bilinear model to the training data, labeled by style (speaker) and content (vowel). We then used the learned vowel parameters $\mathbf{b}^c$ in an SMM and tested classification performance with varying degrees of style information for the 7 new speakers' data:

both style and content labels missing for each test vector (SMM1), style labels present (indicating a change of speaker) but content labels missing (SMM2), and both labels missing but with the test data loglikelihood $\sum_{\mathbf{y}} \log p(\mathbf{y})$ augmented by a prior favoring temporal continuity of style assignments (SMM3).

The few training styles makes this problem difficult and a good showcase for our approach. Robinson [13] obtained 51% correct vowel classification on the test set with a multi-layer perceptron and 56% with a 1-nearest neighbor (1-NN) classifier, the best performance of the many standard techniques he tried. Hastie and Tibshirani [6] recently obtained 62% correct using their discriminant adaptive nearest neighbor algorithm, the best result we know of for an approach that does not model speaker style. We obtained 69% correct for SMM1, 77% for SMM2, and 76% for SMM3, using a model dimensionality of $J = 4$, model variance of $\sigma^2 = .5$, and using the vowel class assignments of 1-NN to initialize the E-step of EM. While good initial conditions were important for the EM algorithm, a range of model dimensionality and variance settings gave reasonable performance.

We also applied these methods to the head pose data of Fig. 2a. We trained on 10 subjects in the 15 poses, and used SMM2 to learn a style model for a new person while simultaneously classifying the head poses. We obtained 81% correct pose categorization (averaged over all 11 test subjects), compared with 53% correct performance for 1-NN matching.

These results demonstrate that modeling style and content can substantially improve content classification in new styles even when no style information is available during testing (SMM1), and dramatically so when some style demarkation is available explicitly (SMM2) or implicitly (SMM3). Bilinear models offer an easy way to improve performance using style labels which are frequently available for many classification tasks.

## 5    Pointers to other work and conclusions

We discuss the extrapolation and translation problems in [2]. Here we summarize results. Figure 3 shows extrapolation of a partially observed font (Monaco) to the unseen letters (see also the gridfont work of [8, 4]). During training, we presented all letters of the five fonts shown at the left. To accomodate many shape topologies, we described letters by the warps of black particles from a reference shape into the letter shape. During testing, we fit an asymmetric model style matrix to all the letters of the Monaco font *except* those shown in the figure. We used the best fitting linear combination of training fonts as a prior for the style matrix, in order to control model complexity. Using the fit style, we then synthesized the unseen letters of the Monaco font. These compare well with the actual letters in that style.

Because the $W$ weights of the symmetric model are independent of any particular style and content class, they allow translation of observations from unknown styles *and* content-classes to known ones. During training, we fit the symmetric model to the observations. For a test observation under a new style and content class, we find $\mathbf{a}^s$ and $\mathbf{b}^c$ values using the known $W$ numbers, iterating least squares fits of the two parameters. Typically, the resulting $\mathbf{a}^s$ and $\mathbf{b}^c$ vectors are unique up to an uncertainty in scale. We have used this approach to translate across shape or lighting conditions for images of faces, and to translate across illumination color for color measurements (assuming small specular reflections).

Our work naturally combines two current themes in the connectionist learning literature: factorial learning [7, 3] and learning a family of many related tasks [1, 12]

Figure 3: Style extrapolation in typography. The training data were all letters of the 5 fonts at left. The test data were all the Monaco letters except those shown at right. The synthesized Monaco letters compare well with the missing ones.

to facilitate task transfer. Separable bilinear models provide a powerful framework for separating style and content by combining explicit representation of each factor with the computational efficiency of linear models.

## Acknowledgements

We thank W. Richards, Y. Weiss, and M. Bernstein for helpful discussions. Joshua Tenenbaum is a Howard Hughes Medical Institute Predoctoral Fellow.

## References

[1] R. Caruana. Learning many related tasks at the same time with backpropagation. In *Adv. in Neural Info. Proc. Systems*, volume 7, pages 657–674, 1995.

[2] W. T. Freeman and J. B. Tenenbaum. Learning bilinear models for two-factor problems in vision. TR 96-37, MERL, 201 Broadway, Cambridge, MA 02139, 1996.

[3] Z. Ghahramani. Factorial learning and the EM algorithm. In *Adv. in Neural Info. Proc. Systems*, volume 7, pages 617–624, 1995.

[4] I. Grebert, D. G. Stork, R. Keesing, and S. Mims. Connectionist generalization for production: An example from gridfont. *Neural Networks*, 5:699–710, 1992.

[5] P. W. Hallinan. A low-dimensional representation of human faces for arbitrary lighting conditions. In *Proc. IEEE CVPR*, pages 995–999, 1994.

[6] T. Hastie and R. Tibshirani. Discriminant adaptive nearest neighbor classification. *IEEE Pat. Anal. Mach. Intell.*, (18):607–616, 1996.

[7] G. E. Hinton and R. Zemel. Autoencoders, minimum description length, and Helmholtz free energy. In *Adv. in Neural Info. Proc. Systems*, volume 6, 1994.

[8] D. Hofstadter. *Fluid Concepts and Creative Analogies*. Basic Books, 1995.

[9] J. J. Koenderink and A. J. van Doorn. The generic bilinear calibration–estimation problem. *Intl. J. Comp. Vis.*, 1997. in press.

[10] J. R. Magnus and H. Neudecker. *Matrix differential calculus with applications in statistics and econometrics*. Wiley, 1988.

[11] D. H. Marimont and B. A. Wandell. Linear models of surface and illuminant spectra. *J. Opt. Soc. Am. A*, 9(11):1905–1913, 1992.

[12] S. M. Omohundro. Family discovery. In *Adv. in Neural Info. Proc. Sys.*, vol. 8, 1995.

[13] A. Robinson. *Dynamic error propagation networks*. PhD thesis, Cambridge University Engineering Dept., 1989.

[14] C. Tomasi and T. Kanade. Shape and motion from image streams under orthography: a factorization method. *Intl. J. Comp. Vis.*, 9(2):137–154, 1992.

[15] M. Turk and A. Pentland. Eigenfaces for recognition. *J. Cog. Neurosci.*, 3(1), 1991.